# Learning Hyper-Features for Visual Identification

**Andras Ferencz**     **Erik G. Learned-Miller**     **Jitendra Malik**
Computer Science Division, EECS
University of California at Berkeley
Berkeley, CA 94720

## Abstract

We address the problem of identifying specific instances of a class (cars) from a set of images all belonging to that class. Although we cannot build a model for any particular instance (as we may be provided with only one "training" example of it), we can use information extracted from observing other members of the class. We pose this task as a learning problem, in which the learner is given image pairs, labeled as matching or not, and must discover which image features are most consistent for matching instances and discriminative for mismatches. We explore a patch based representation, where we model the distributions of similarity measurements defined on the patches. Finally, we describe an algorithm that selects the most salient patches based on a mutual information criterion. This algorithm performs identification well for our challenging dataset of car images, after matching only a few, well chosen patches.

## 1 Introduction

Figure 1 shows six cars: the two leftmost cars were captured by one camera; the right four cars were seen later by another camera from a different angle. The goal is to determine which images, if any, show the *same vehicle*. We call this task *visual identification*. Most existing identification systems are aimed at biometric applications such as identifying fingerprints or faces. While *object recognition* is used loosely for several problems (including this one), we differentiate visual *identification*, where the challenge is distinguishing between visually similar objects of one category (e.g. faces, cars), and *categorization* where

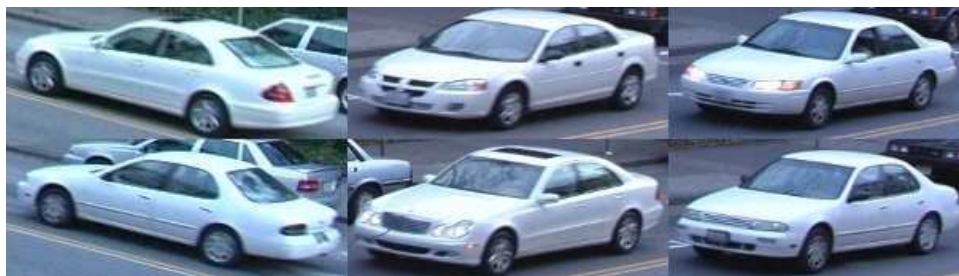

Figure 1: *The Identification Problem: Which of these cars are the same?* The two cars on the left, photographed from camera 1, also drive past camera 2. Which of the four images on the right, taken by camera 2, match the cars on the left? Solving this problem will enable applications such as wide area tracking of cars with a sparse set of cameras [2, 9].

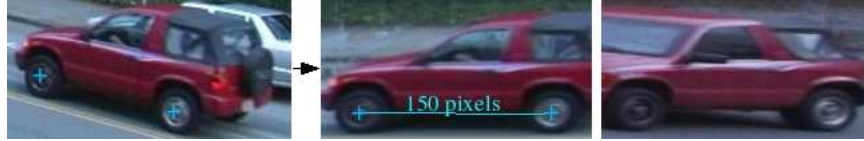

Figure 2: *Detecting and warping car images into alignment:* Our identification algorithm assumes that a detection process has found members of the class and approximately aligned them to a canonical view. For our data set, detection is performed by a blob tracker. A projective warp to align the sides is computed by calibrating the pose of the camera to the road and finding the wheels of the vehicle. Note that this is only a rough approximation (the two warped images, center and right, are far from perfectly aligned) that helps to simplify our patch descriptors and positional bookkeeping.

the algorithm must group together objects that belong to the same category but may be visually diverse[1, 5, 10, 13]. Identification is also distinct from "*object localization*," where the goal is locating a specific object in scenes in which distractors have little similarity to the target object [6].[1]

One characteristic of the identification problem is that the algorithm typically only receives one positive example of each query class (e.g. a single image of a specific car), before having to classify other images as the "same" or "different". Given this lack of a class specific training set, we cannot use standard supervised feature selection and classification methods such as [12, 13, 14]. One possible solution to this problem is to try to pick universally good features, such as corners [4, 6], for detecting salient points. However, such features are likely to be suboptimal as they are not category specific. Another possibility is to hand-select good features for the task, such as the distance between the eyes for face identification.

Here we present an identification framework that attempts to be more general. The core idea is to use a training set of other image pairs from the category (in our case cars), labeled as matching or not, to learn what characterizes features that are informative in distinguishing one instance from another (i.e. consistent for matching instances and dissimilar for mismatches). Our algorithm, given a single novel query image, can build a "same" vs. "different" classifier by: (1) examining a set of candidate features (local image patches) on the query image (2) selecting a small number of them that are likely to be the most informative for this query class and (3) estimating a function for scoring the match for each selected feature. Note that a different set of features (patches) will be selected for each unique query.

The paper is organized as follows. In Section 2, we describe our decision framework including the decomposition of an image pair into *bi-patches*, which give local indications of match or mismatch, and introduce the appearance distance between the two halves as a discriminative statistic of bi-patches. This model is then refined in Section 3 by conditioning the distance distributions on hyper-features such as patch location, contrast, and dominant orientation. A patch saliency measure based on the estimated distance distributions is introduced in Section 3.4. In Section 4, we extend our model to include another comparison statistic, the difference in patch position between images. Finally, in Section 5, we conclude and show that comparing a small number of well-chosen patches produces performance nearly as good as matching a dense sampling of them.

## 2   Matching Patches

We seek to determine whether a new query image $I^L$ (the "Left" image) represents the same vehicle as any of our previously seen database images $I^R$ (the "Right" image). We assume that these images are known to contain vehicles, have been brought into rough correspondence (in our data set, through a projective transformation that aligns the sides of the car) and have been scaled to approximately 200 pixels in length (see Figure 2 for details).

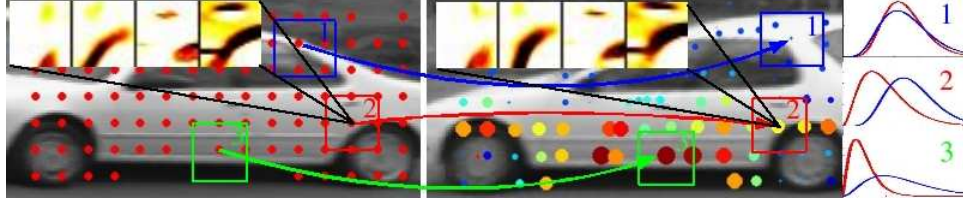

Figure 3: *Patch Matching:* The left (query) image is sampled (red dots) by patches encoded as oriented filter channels (for labeled patch 2, this encoding is shown). Each patch is matched to the best point in the database image of the same car by maximizing the appearance similarity between the patches (the similarity score is indicated by the size and color of the dots, where larger and redder is more similar). Three bi-patches are labeled. Although the classification result for this pair of images should be "same" ($C = 1$), notice that some bi-patches are better predictors of this result than others (the similarity score of 2 & 3 is much better than for patch 1). Our goal is to be able to predict the distribution of $P(d|C = 1)$ and $P(d|C = 0)$ for each patch accurately based on the appearance and position of the patch in the query image (for the 3 patches, our predictions are shown on the right).

## 2.1 Image Patch Features

Our strategy is to break up the whole image comparison problem into multiple local matching problems, where we encode a small patch $F_j^L$ ($1 \leq j \leq n$) of the query image $I^L$ and compare each piece separately [12, 14]. As the exact choice of features, their encoding and comparison metric is not crucial to our technique, we chose a fairly simple representation that was general enough to use in a wide variety of settings, but informative enough to capture the details of objects (given the subtle variation that can distinguish two different cars, features such as [6] were found not to be precise enough for this task).

Specifically, we apply a first derivative Gaussian odd-symmetric filter to the patch at four orientations (horizontal, vertical, and two diagonal), giving four signed numbers per pixel. To compare a query patch $F_j^L$ to an area of the right image $F_j^R$, we encode both patches as $4 \times 25^2$ length vectors (4 orientations per pixel) and compute the normalized correlation ($d_j = 1 - CorrCoef(F_j^L, F_j^R)$) between these vectors. As the two car images are in rough alignment, we need only to search a small area of $I^R$ to find the best corresponding patch $F_j^R$ - i.e. the one that minimizes $d_j$. We will refer to such a matched left and right patch pair $F_j^L, F_j^R$, together with the derived distance $d_j$, as a bi-patch $F_j$.

## 2.2 The Decision Rule

We pose the task of deciding if the a database image $I^R$ is the same as a query image $I^L$ as a decision rule

$$R = \frac{P(C = 1|I^L, I^R)}{P(C = 0|I^L, I^R)} = \frac{P(I^L, I^R|C = 1)P(C = 1)}{P(I^L, I^R|C = 0)P(C = 0)} > \lambda. \quad (1)$$

where $\lambda$ is chosen to balance the cost of the two types of decision errors. The priors are assumed to be known.[2] Specifically, for the remaining equations in this paper, the priors are assumed to be equal, and hence are dropped from subsequent equations. With our image decomposition into patches, the posteriors from Eq. (1) will be approximated using the bi-patches $F_1, ..., F_n$ as $P(C|I^L, I^R) \approx P(C|F_1, ..., F_m) \propto P(F_1, ..., F_m|C)$. Furthermore, in this paper, we will assume a naive Bayes model in which, conditioned on $C$, the bi-patches are assumed to be independent. That is,

$$R = \frac{P(I^L, I^R|C = 1)}{P(I^L, I^R|C = 0)} \approx \frac{P(F_1, ..., F_m|C = 1)}{P(F_1, ..., F_m|C = 0)} = \prod_{j=1}^{m} \frac{P(F_j|C = 1)}{P(F_j|C = 0)}. \quad (2)$$

In practice, we compute the log of this likelihood ratio, where each patch contributes an additive term (denoted $\mathcal{LLR}_i$ for patch $i$). Modeling the likelihoods in this ratio ($P(F_j|C)$) is the central focus of this paper.

## 2.3 Uniform Appearance Model

The most straightforward way to estimate $P(F_j|C)$ is to assume that the appearance difference $d_j$ captures all of the information $F_j$ about the probability of a match (i.e. $C$ and $F_j$ are independent given $d_j$), and that all of $d_j$'s from all patches are identically distributed. Thus the decision rule, Eqn. 1, becomes

$$R \approx \prod_{j=1}^{m} \frac{P(d_j|C=1)}{P(d_j|C=0)} > \lambda. \quad (3)$$

The two conditional distributions, $P(d_j\,|\,C \in \{0,1\})$, are estimated as normalized histograms from all bi-patches matched within the training data.[3] For each value of $\lambda$, we evaluate Eqn.(3) to classify each test pair as matching or not, producing a precision-recall curve. Figure 4 compares this *patch-based* model to a *direct image comparison* method.[4] Notice that even this naive patch-based technique significantly outperforms the global matching.

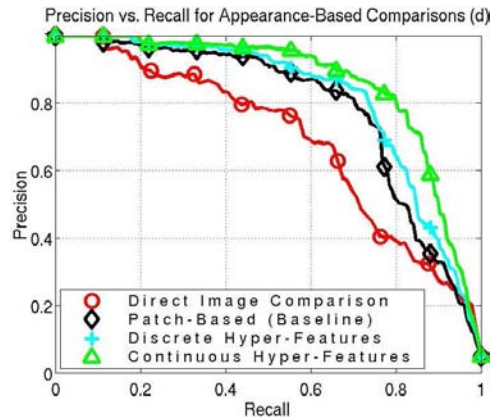

Figure 4: *Identification using appearance differences:* The bottom curve shows the precision vs. recall for non-patch based direct comparison of rectified images. (An ideal precision-recall curve would reach the top right corner.) Notice that all three patch based models outperform this method. The three top curves show results for various models of $d_j$ from Sections 2.3 (Baseline), 3.1 (Discrete), and 3.2 & 3.3 (Continuous). The regression model outperforms the uniform one significantly - it reduces the error in precision by close to 50% for most values of recall below 90%.

## 3 Refining the Appearance Distributions with Hyper-Features

The most significant weakness of the above model is the assumption that the $d_j$'s from different bi-patches should be identically distributed (observe the 3 labeled patches in Figure 3). When a training set of "same" ($C=1$) and "different" ($C=0$) images is available *for a specific query image*, estimating these distributions directly for each patch is straightforward. How can we estimate a distribution for $P(d_j|C=1)$, where $F_j^L$ is a patch from a new query image, when we only have that *single positive example* of $F_j^L$? The intuitive answer: by finding analogous patches in the training set of labeled (same/different) image pairs. However, since the space of all possible patches (appearance & position, $\Re^{25*25+2}$) is very large, the chance of having seen a very similar patch to $F_j^L$ in the training set is small. In the next sections we present two approaches both of which rely on projecting $F_j^L$ into a much lower dimensional space by extracting meaningful features from its position and appearance (the *hyper-features*).

### 3.1 Non-Parametric Model with Discrete Hyper-Features

First we attempted a non-parametric approach, where we model the joint distribution of $d_j$ and a few hyper-features (e.g. the x and y coordinate of the patch $F_j^L$,

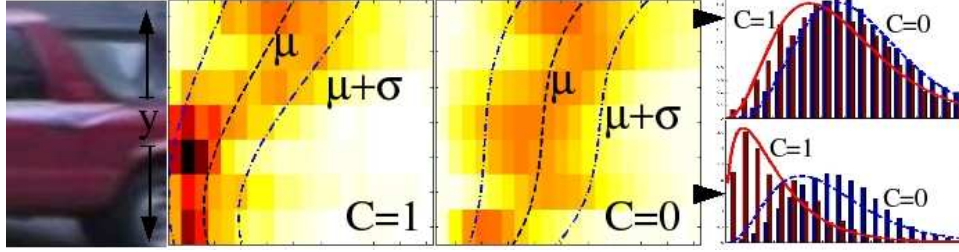

Figure 5: *Fitting a GLM to the $\Gamma$ distribution:* we demonstrate our approach by fitting a gamma distribution, through the latent variables $\Theta = (\mu, \gamma)$, to the y position of the patches. Here we allowed $\mu$ and $\sigma$ to be a 3rd degree polynomial function of y (i.e. $\mathbf{Z} = [\mathbf{y^3, y^2, y, 1}]^{\mathbf{T}}$). The center-left square shows, on each row, a distribution of $d$ conditioned on the $y$ position of the left patch ($F^L$) for each bi-patch, for training data taken from matching vehicles. The center-right square shows the same distributions for mismatched data. The height of histogram distributions is color-coded, dark red indicating higher density. The central curve shows the polynomial fit to the conditional means, while the outer curves show the $\pm\sigma$ range. For reference, we include a partial image of a car whose y-coordinate is aligned with the center images. On the right, we show two histogram plots, each corresponding to one row of the center images (a small range of y corresponding to the black arrows). The resulting gamma distributions are superimposed on the histograms.

i.e. $Z = [x, y]$). The distribution is modeled "non-parametrically" (similar to Section 2.3) using an N-dimensional normalized histogram where each dimension ($d, x$, and $y$) has been quantized into several bins. In this model $P(C|F_j) \approx P(C|d_j, y_j, x_j) \propto P(d_j|y_j, x_j, C)P(y_j, x_j|C)P(C) \propto P(d_j|y_j, x_j, C)$, where the last formula follows from the assumption of equal priors ($P(C) = 0.5$) and the independence of $(y_j, x_j)$ and $C$. The *Discrete Hyper-Features* curve in Figure 4 shows the performance gain from conditioning on these positional hyper-features.

## 3.2 Parametric Model with Continuous Hyper-Features

The drawback of using a non-parametric model for the distributions is that the amount of data needed to populate the histograms grows exponentially with the number of dimensions. In order to add additional appearance-based hyper-features, such as contrast, oriented edge energy, etc., we moved to a smooth parametric representation for both the distribution of $d_j$ and the model by which the the hyper-features influence this distribution.

Specifically, we model the distributions $P(d_j|C = 1)$ and $P(d_j|C = 0)$ as gamma distributions (notated $\Gamma()$) parameterized by the mean and shape parameter $\theta = \{\mu, \gamma\}$ (see the right panel of Figure 5 for examples of the $\Gamma()$ fitting the empirical distributions). The smooth variation of $\theta$ with respect to the hyper-features can be modeled using a generalized linear model (GLM). Ordinary (least-squares) linear models assume that the data is normally distributed with constant variance. GLMs are extensions to ordinary linear models that can fit data which is not normally distributed and where the dispersion parameter also depends on the covariates (see [7] for more information on GLMs).

Our goal is to fit gamma distributions to the distributions of $d$ values for various patches by maximizing the probability density of data under gamma distributions whose parameters are simple polynomial functions of the hyper-features. Consider a set $X_1, ..., X_k$ of hyper-features such as position, contrast, and brightness of a patch. Let $\mathbf{Z} = [Z_1, ..., Z_l]^T$ be a vector of $l$ pre-chosen functions of those hyper-features, like squares, cubes, cross terms, or simply copies of the variables themselves. Then each bi-patch distance distribution has the form

$$P(d|X_1, X_2, ..., X_k, C) = \Gamma(d; \ \alpha_{\mathbf{C}}^{\mu} \cdot \mathbf{Z}, \ \alpha_{\mathbf{C}}^{\gamma} \cdot \mathbf{Z}), \tag{4}$$

where the second and third arguments to $\Gamma()$ are mean and shape parameters.[5] Each $\alpha$ (there are four of these: $\alpha_{C=0}^{\mu}, \alpha_{C=0}^{\gamma}, \alpha_{C=1}^{\mu}, \alpha_{C=1}^{\gamma}$) is a vector of parameters of length $l$

that weights each hyper-feature monomial $\mathbf{Z}_i$. The $\alpha$'s are adapted to maximize the joint data likelihood over all patches for $C = 0$ or $C = 1$ withing the training set. These ideas are illustrated in detail in Figure 5.

### 3.3 Automatic Selection of Hyper-Features

In this section we describe the automatic determination of $\mathbf{Z}$. Recall that in our GLM model we assumed a linear relationship between $\mathbf{Z}$ and $\mu, \gamma$. This allows us to use standard feature selection techniques, such as Least Angle Regression (LARS)[3], to choose a few (around 10) hyper-features from a large set of candidates,[6] such as: (a) the x and y positions of $F^L$, (b) the intensity and contrast within $F^L$ and the average intensity of the entire vehicle, (c) the average energy in each of the 8 oriented filter channels, and (d) derived quantities from the above (e.g. square, cubic, and cross terms). LARS was then asked to choose $\mathbf{Z}$ from these features. Once $\mathbf{Z}$ is set, we proceed as in Section 3.2.

Running an automatic feature selection technique on this large set of possible conditioning features gives us a principled method of reducing the complexity of our model. Reducing the complexity is important not only to speed up computation, but also to mitigate the risk of over-fitting to the training set. The top curve in Figure 4 shows results when $\mathbf{Z}$ includes the first 10 features found by LARS. Even with such a naive set of features to choose from, the performance of the system improves significantly.

### 3.4 Estimating the Saliency of a Patch

From the distributions $P(d_j|C = 0)$ and $P(d_j|C = 1)$ computed separately for each patch, it is also possible to estimate the saliency of the patch, i.e. the amount of information about our decision variable $C$ we are likely to gain should we compute the best corresponding $F_j^R$. Intuitively, if the distribution of $D_j$ is very different for $C = 0$ and $C = 1$, then the amount of information gained by matching patch $j$ is likely to be large (see the 3 distributions on the right of Figure 3). To emphasize the fact that the distribution $P(d_j|C)$ is a fixed function of $F_j^L$, given the learned hyper-feature weights $\alpha$, we slightly abuse notation and refer to the random variable from which $d_j$ is sampled as $F_j^L$.

With this notation, computing the mutual information between $F_j^L$ and $C$ gives us a measure of the expected information gain from a patch with particular hyper-features:

$$I(F_j^L; C) = H(F_j^L) - H(F_j^L|C).$$

Here $H()$ is Shannon entropy. The key fact to notice is that this measure can be computed just from the estimated distributions over $d_j$ (which, in turn, were estimated from the hyper-features of $F_j^L$) before the patch has been matched. This allows us to match only those patches that are likely to be informative, leading to significant computational savings.

## 4 Modeling Appearance and Position Differences

In the last section, we only considered the similarity of two matching patches that make up a bi-patch in terms of the *appearance* of the patches ($d_j$). Recall that for each left patch $F_j^L$, a matching right patch $F_j^R$ is found by searching for the most similar patch in some large neighborhood around the expected location for the match. In this section, we show how to model the change in *position*, $r_j$, of the match relative to its expected location, and how this, when combined with the appearance model, improves the matching performance.

---

solving an ordinary least squares problem. We experimentally compared it to the canonical inverse link ($\mu = (\alpha_C^{\mu}{}^T * \mathbf{Z})^{-1}$), but observed no noticeable change in performance on our data set.

[6]In order to use LARS (or most other feature selection methods) "out of the box", we use regression based on an $L2$ loss function. While this is not optimal for non-normal data, from experiments we have verified that it is a reasonable approximation for the feature selection step.

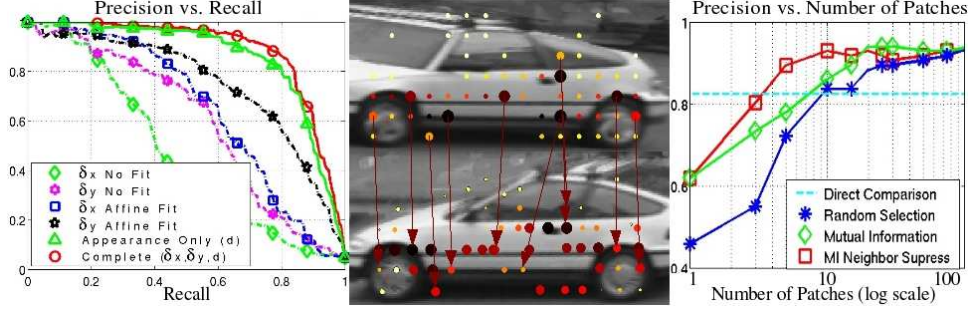

Figure 6: *Results:* The **LEFT** plot shows precision vs. recall curves for models of $r$. The results for $\delta x$ and $\delta y$ are shown separately (as there are often more horizontal than vertical features on cars, $\delta y$ is better). Re-estimating parameters of the global alignment, $W$ (affine fit), significantly improves the curves. Finally, performance is improved by combining position with appearance ("Complete" curve) compared to using appearance alone. The **CENTER** pair of images show a correct match, with the patch centers indicated by circles. The **color of the circles** in the top image indicates $\mathcal{MI}_j$, in bottom image $\mathcal{LLR}_j$. Our patch selection algorithm chooses the top patches based on $\mathcal{MI}$ where subsequent patches are penalized for overlapping with earlier ones (neighborhood suppression). The top 10 "left" patches chosen are marked with **arrows** connecting them to the corresponding "right" patches. Notice that these are concentrated in informative regions. The **RIGHT** plot quantifies this observation: the curves show 3 different methods of choosing the order of patches - random order, $\mathcal{MI}$ and $\mathcal{MI}$ with neighborhood suppression. Notice that this top curve with 3 patches does as well as the direct comparison method. All 3 methods converge above 50 patches.

Let $r_j = (\delta x_j, \delta y_j)$ be the difference in position between the coordinates of $F_j^L$ and $F_j^R$ within the standardized coordinate frames. Generally, we expect $r_j \approx 0$ if the two images portray the same object ($C = 1$). The estimate for $R$, incorporating the information from both $d$ and $r$ becomes

$$R \approx \prod_{j=1}^{m} \frac{P(r_j|d_j, \mathbf{Z}_j, C=1)P(d_j|\mathbf{Z}_j, C=1)}{P(r_j|d_j, \mathbf{Z}_j, C=0)P(d_j|\mathbf{Z}_j, C=0)}, \tag{5}$$

where $\mathbf{Z}_j$ again refers to a set of hyper-features.

Here we focus on the first factor, where the distribution of $r_j$ given $C$ is dependent on the appearance and position of the left patch ($F_j^L$, through the hyper-features $\mathbf{Z}_j$) and on the similarity in appearance ($d_j$). The intuition for the dependence on $d_j$ is that for the $C = 1$ case, we expect $r_j$ to be smaller on average when a good appearance match (small $d_j$) was found.

Following our approach for $d_j$, we model the distribution of $r_j$ as a 0 mean normal distribution, $\mathcal{N}(0, \Sigma)$, where $\Sigma$ (we use a diagonal covariance) is a function of $\mathbf{Z}_j, d_j$. The parameterization of $(\mathbf{Z}_j, d_j)$ is found through feature selection, while the weights for the linear function are obtained by maximizing the likelihood of $r_j$ over the training data. To address initial misalignment, we select a small number of patches, match them, and compute a global affine alignment between the images. We subsequently score each match relative to this global alignment.

The bottom four curves of Figure 6 show that fitting an affine model first significantly improves the positional signal. While position seems to be less informative than appearance, the complete model, which combines appearance and position (Eq. 5), outperforms appearance alone.

## 5 Conclusion

The center and right sides of Figure 6 show our ability to select the most informative patches using the estimated mutual information $I(F_j^L, C)$ of each patch. To prevent spatially overlapping patches from being chosen, we added a penalty factor to the mutual in-

formation score that penalizes patches that are very close to other chosen patches (MI with neighborhood suppression). To give a numerical indication of the performance, we note that with only 10 patches, given a 1-to-87 forced choice problem, our algorithm chooses the correct matching image 93% of the time.

A different approach to a learning problem that is similar to ours can be found in [5, 8], which describe methods for learning character or object categories from few training examples. These works approach this problem by learning distributions on shared factors [8] or priors on parameters of fixed distributions for a category [5] where the training data consists of images from other categories. We, on the other hand, abandon the notion of building a model with a fixed form for an object from a single example. Instead, we take a discriminative approach and model the statistical properties of image patch differences conditioned on properties of the patch. These learned conditional distributions allow us to evaluate, for each feature, the amount of information potentially gained by matching it to the other image.[7]

## Acknowledgments

This work was partially funded by DARPA under the *Combat Zones That See* project.

## Footnotes

[1]There is evidence that this distinction exists in the human visual system. Some findings suggest that the fusiform face area is specialized for identification of instances from familiar categories[11].

[2]For our application, dynamic models of traffic flow can supply the prior on $P(C)$.

[3]Data consisted of 175 pairs (88 training, 87 test pairs) of matching car images (C=1) from two cameras located on the same side of the street one block apart. Within training and testing sets, about 4000 pairs of mismatched cars (C=0) were formed from non-corresponding images, one from each camera. All comparisons were performed on grayscale (not color) images.

[4]The global image comparison method used here as a baseline technique uses normalized correlation on a combination of intensity and filter channels, and attempts to overcome slight misalignment.

[5]For the GLM, we use the identity link function for both $\mu$ and $\gamma$. While the identity is not the canonical link function for $\mu$, its advantage is that our ML optimization can be initialized by

[7]Answer to Figure 1: top left matches bottom center; bottom left matches bottom right. For our algorithm, matching these images was not a challenge.

## References

[1] Y. Amit and D. Geman. A computational model for visual selection. *Neural Computation*, 11(7), 1999.

[2] D. Beymer, P. McLauchlan, B. Coifman, and J. Malik. A real-time computer vision system for measuring traffic parameters. *CVPR*, 1997.

[3] B. Efron, T. Hastie, I. Johnstone, and R. Tibshirani. Least angle regression. *Annals of Statistics*, 32(2):407–499, 2004.

[4] T. Kadir and M. Brady. Scale, saliency and image description. *International Journal of Computer Vision*, 45(2):83–105, 2001.

[5] F. Li, R. Fergus, and P. Perona. A Bayesian approach to unsupervised one-shot learning of object categories. In *ICCV*, 2003.

[6] D. Lowe. Distinctive image features from scale-invariant keypoints. *International Journal of Computer Vision*, 60(2):91–110, 2004.

[7] P. McCullagh and J. A. Nelder. *Generalized Linear Models*. Chapman and Hall, 1989.

[8] E. Miller, N. Matsakis, and P. Viola. Learning from one example through shared densities on transforms. In *CVPR*, 2000.

[9] H. Pasula, S. Russell, M. Ostland, and Y. Ritov. Tracking many objects with many sensors. *IJCAI*, 1999.

[10] H. Schneiderman and T. Kanade. A statistical approach to 3d object detection applied to faces and cars. *CVPR*, 2000.

[11] M. Tarr and I. Gauthier. FFA: A flexible fusiform area for subordinate-level visual processing automatized by expertise. *Nature Neuroscience*, 3(8):764–769, 2000.

[12] M. Vidal-Naquet and S. Ullman. Object recognition with informative features and linear classification. In *International Conference on Computer Vision*, 2003.

[13] P. Viola and M. Jones. Rapid object detection using a boosted cascade of simple features. In *CVPR*, 2001.

[14] M. Weber, M. Welling, and P. Perona. Unsupervised learning of models for recognition. *ECCV*, 2000.

